# Speech Recognition with Missing Data using Recurrent Neural Nets

**S. Parveen**
Speech and Hearing Research Group
Department of Computer Science
University of Sheffield
Sheffield S14DP, UK
*s.parveen@dcs.shef.ac.uk*

**P.D. Green**
Speech and Hearing Research Group
Department of Computer Science
University of Sheffield
Sheffield S14DP, UK
*p.green@dcs.shef.ac.uk*

## Abstract

In the 'missing data' approach to improving the robustness of automatic speech recognition to added noise, an initial process identifies spectral-temporal regions which are dominated by the speech source. The remaining regions are considered to be 'missing'. In this paper we develop a connectionist approach to the problem of adapting speech recognition to the missing data case, using Recurrent Neural Networks. In contrast to methods based on Hidden Markov Models, RNNs allow us to make use of long-term time constraints and to make the problems of classification with incomplete data and imputing missing values interact. We report encouraging results on an isolated digit recognition task.

## 1. Introduction

Automatic Speech Recognition systems perform reasonably well in controlled and matched training and recognition conditions. However, performance deteriorates when there is a mismatch between training and testing conditions, caused for instance by additive noise (Lippmann, 1997). Conventional techniques for improving recognition robustness (reviewed by Furui 1997) seek to eliminate or reduce the mismatch, for instance by enhancement of the noisy speech, by adapting statistical models for speech units to the noise condition or simply by training in different noise conditions.

Missing data techniques provide an alternative solution for speech corrupted by additive noise which make minimal assumptions about the nature of the noise. They are based on identifying uncorrupted, reliable regions in the frequency domain and adapting recognition algorithms so that classification is based on these regions.

Present missing data techniques developed at Sheffield (Barker et al. 2000a, Barker et al. 2000b, Cooke et al., 2001) and elsewhere (Drygaglo et al., 1998, Raj et al., 2000) adapt the prevailing technique for ASR based on Continuous Density Hidden Markov Models. CDHMMs are generative models which do not give direct estimates of posterior

probabilities of the classes given the acoustics. Neural Networks, unlike HMMs, are discriminative models which do give direct estimates of posterior probabilities and have been used with success in hybrid ANN/HMM speech recognition systems (Bourlard et al., 1998).

In this paper, we adapt a recurrent neural network architecture introduced by (Gingras & Bengio, 1998) for robust ASR with missing data.

## 2. Missing data techniques for Robust ASR

### 2.1 Missing data masks

Speech recognition with missing data is based on the assumption that some regions in time/frequency remain uncorrupted for speech with added noise. See (Cooke et al., 2001) for arguments to support this assumption. Initial processes, based on local signal-to-noise estimates, on auditory grouping cues, or a combination (Barker et al., 2001) define a binary 'missing data mask': ones in the mask indicate reliable (or 'present') features and zeros indicate unreliable (or 'missing') features.

### 2.2 Classification with missing data

Techniques for classification with incomplete data can be divided into imputation and marginalisation. *Imputation* is a technique in which missing features are replaced by estimated values to allow the recognition process proceed in normal way. If the missing values are replaced by either zeros, random values or their means based on training data, the approach is called *unconditional imputation.* On the other hand in *conditional imputation* conditional statistics are used to estimate the missing values given the present values. In the *marginalisation* approach missing values are ignored (by integrating over their possible ranges) and recognition is performed with the reduced data vector which is considered reliable. For the multivariate mixture Gaussian distributions used in CDHMMs, marginalisation and conditional imputation can be formulated analytically (Cooke et al., 2001). For missing data ASR further improvements in both techniques follow from using the knowledge that for spectral energy features the unreliable data is bounded between zero and the energy in speech+noise mixture (Vizinho et al., 1999), (Josifovski et al., 1999). These techniques are referred to as *bounded marginalisation* and *bounded imputation.* Coupled with a 'softening' of the reliable/unreliable decision, missing data techniques produce good results on a standard connected-digits-in-noise recognition task: performance using models trained on clean data is comparable, and in severe noise superior, to conventional systems trained across different noise conditions (Barker et al., 2001).

### 2.3 Why recurrent neural nets for missing data robust ASR?

Several neural net architectures have been proposed to deal with the missing data problem in general (Ahmed & Tresp, 1993), (Ghahramani & Jordan, 1994). The problem in using neural networks with missing data is to compute the output of a node/unit when some of its input values are unavailable.

For marginalisation, this involves finding a way of integrating over the range of the missing values. A robust ASR system to deal with missing data using neural networks has recently been proposed by (Morris et al., 2000). This is basically a radial basis function neural network with the hidden units associated with a diagonal covariance gaussian. The marginal over the missing values can be computed in this case and hence the resulting system is equivalent to the HMM based missing data speech recognition system using marginalisation. Reported performance is also comparable to that of the HMM based

speech recognition system.

In this paper missing data is dealt with by imputation. We use recurrent neural networks to estimate missing values in the input vector. RNNs have the potential to capture long-term contextual effects over time, and hence to use temporal context to compensate for missing data which CDHMM based missing data techniques do not do. The only contextual information available in CDHMM decoding come from the addition of temporal derivatives to the feature vector. RNNs also allow a single net to perform both imputation and classification, with the potential of combining these processes to mutual benefit.

The RNN architecture proposed by Gingras et al. (1998) is based on a fully-connected feedforward network with input, hidden and output layers using hyperbolic tangent activation functions. The output layer has one unit for each class and the network is trained with the correct classification as target. Recurrent links are added to the feedforward net with unit delay from output to the hidden units as in Jordan networks (Jordan, 1988). There are also recurrent links with unit delay from hidden units to missing input units to impute missing features. In addition, there are self delayed terms with a fixed weight for each unit which basically serve to stabilise RNN behaviour over time and help in imputation as well. Gingras et al. used this RNN both for a pattern classification task with static data (one input vector for each example) and sequential data (a sequence of input values for each example).

Our aim is to adapt this architecture for robust ASR with missing data. Some preliminary static classification experiments were performed on vowel spectra (individual spectral slices excised from the TIMIT database). RNN performance on this task with missing data was better than standard MLP and gaussian classifiers. In the next section we show how the net can be adapted for dynamic classification of the spectral sequences constituting words.

## 3. RNN architecture for robust ASR with missing data

Figure 1 illustrates our modified version of the Gingras and Bengio architecture. Instead of taking feedback from the output to the hidden layer we have chosen a fully connected or Elman RNN (Elman, 1990) where there are full recurrent links from the past hidden layer to the present hidden layer (figure 1). We have observed that these links produce faster convergence, in agreement with (Pedersen, 1997). The number of input units depends on the size of feature vector, i.e. the number of spectral channels. The number of hidden units is determined by experimentation. There is one output unit for each pattern class. In our case the classes are taken to be whole words, so in the isolated digit recognition experiments we report, there are eleven output units, for '1' - '9', 'zero' and 'oh'.

In training, missing inputs are initialised with their unconditional means. The RNN is then allowed to impute missing values for the next frame through the recurrent links, after a feedforward pass.

$$X_{(m,t)} = (1 - \gamma)X_{(m,t-1)} + \sum_{j=1}^{H} v_{jm} f(hid_{(j,t-1)})$$

Where $X_{(m,t)}$ is the missing feature at time $t$, $\gamma$ is the learning rate, $v_{jm}$ indicates recurrent links from a hidden unit to the missing input and $hid_{(j,t-1)}$ is the activation of hidden unit $j$ at time $t$-1.

The average of the RNN output over all the frames of an example is taken after these frames have gone through a forward pass. The sum squared error between the correct targets and the RNN output for each frame is back-propagated through time and RNN

weights are updated until a stopping criterion is reached.

The recognition phase consists of a forward pass to produce RNN output for unseen data and imputation of missing features at each time step. The highest value in the averaged output vector is taken as the correct class.

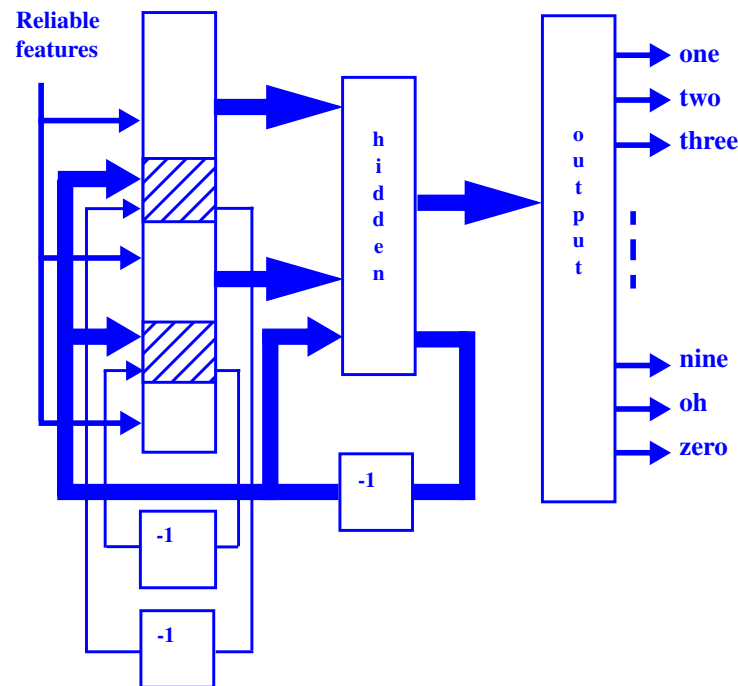

Figure 1: RNN architecture for robust ASR with missing data technique. Solid arrows show full forward and recurrent connections between two layers. Shaded blocks in the input layer indicate missing inputs which keep changing at every time step. Missing inputs are fully connected (solid arrows) with the hidden layer with a unit delay in addition to delayed self-connection (thin arrows) with a fixed weight.

## 4. Isolated word recognition experiments

Continuous pattern classification experiments were performed using data from 30 male speakers in the isolated digits section of the TIDIGIT database (Leonard, 1984). There were two examples per speaker of each of the 11 words (i.e. 1-9, zero, oh). 220 examples were chosen from a subset of 10 speakers for training. Recognition was performed on 110 examples from the speakers not included in training. A validation set of 110 examples was used for early stopping.

Features were extracted from hamming windowed speech with a window size of 25 msec and 50% overlap. Two types of feature vectors used for the experiments were total energies in the four frequency bands (115-629 Hz, 565-1370 Hz, 1262-2292 Hz and 2212-3769 Hz) and 20 mel scaled FFT filter bank energies.

In the initial experiments we report, the missing data masks were formed by deleting

spectral energy features at random. This allows comparison with early results with HMM-based missing data recognition (Cooke et al. 1996) and close experimental control. For training 1/3rd of the training examples were clean, 1/3rd had 25% deletions and 1/3rd had 50% deletions. Recognition performance was evaluated with 0% to 80% missing features with an increment of 10%.

## 5. Results

### 5.1 RNN performance as a classifier

An RNN with 20 inputs, 65 hidden and 11 output units was chosen for recognition and imputation with 20 features per time frame. Its performance on various amounts of missing features from 0% to 80%, shown in Figure 2 (the 'RNN imputation' curve), is much better than the standard Elman RNN trained on clean speech only for classification task and tested with the mean imputation. Use of the self delayed term in addition to the recurrent links for imputation of missing features contributes positively in case of sequential data. Results resemble those reported for HMMs in (Cooke et al. 1996). We also show that results are superior to 'last reliable imputation' in which the imputed value of a feature is the last reliable value for that feature.

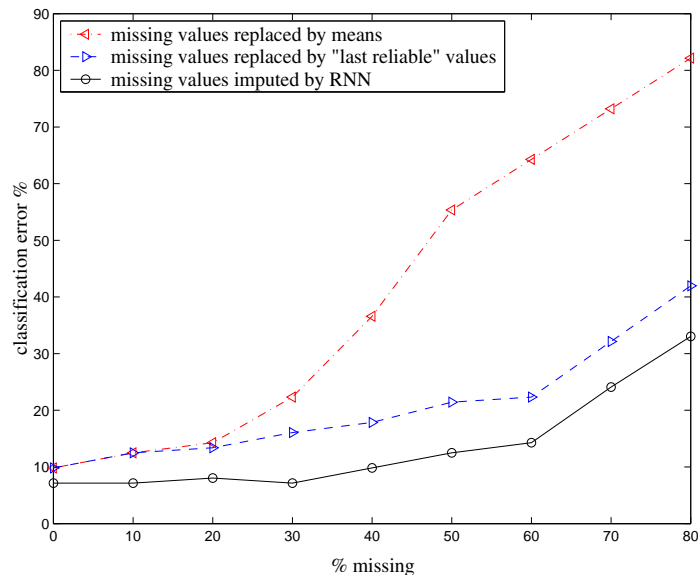

Figure 2: Comparison of RNN classification performance for different imputation methods.

### 5.2 RNN performance on pattern completion

Imputation, or pattern completion, performance was observed for an RNN trained with 4 features per frame of the speech and is shown in Figure 3. The RNN for this task had 4 input, 45 hidden and 11 output units. In figure 3(a), solid curves show the true values of the feature in each frequency band at every frame for an example of a spoken '9', the horizontal lines are mean feature values, and the circles are the missing values imputed by the RNN. Imputed values are encouragingly close to the true values. For this network, classification error for recognition was 10.7% at 0% missing and 46.4% at 80% missing.

The top curve in figure 3(b) shows the average pattern completion error when an RNN with 20 input channels was trained on clean speech and during recognition missing features were imputed from their unconditional means. The bottom curve is the average pattern completion error with missing features imputed by the network. This demonstrates the clear advantage of using the RNN for both imputation and classification.

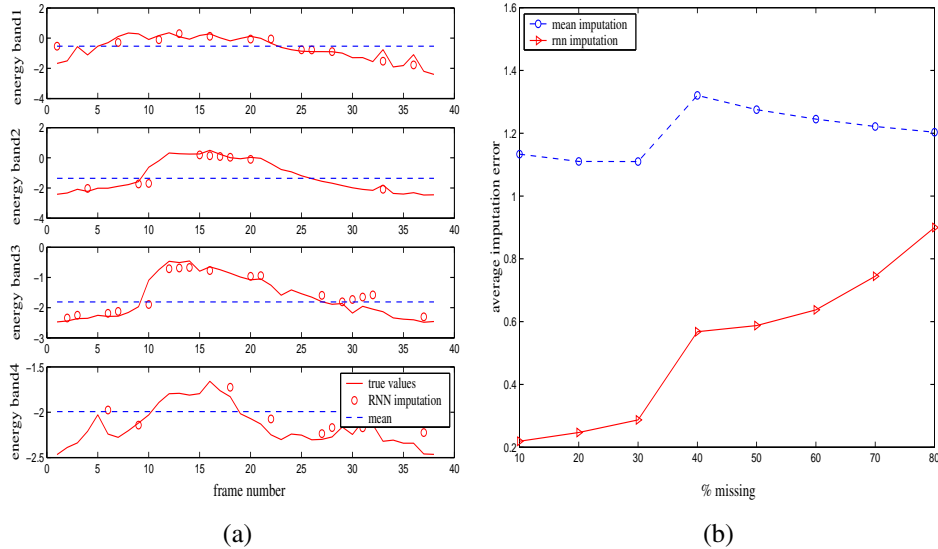

(a) (b)

Figure 3: (a) Missing values for digit 9 imputed by an RNN (b) Average imputation errors for mean imputation and RNN imputation

## 6. Conclusion & future work

The experiments reported in section 5 constitute no more than a proof of concept. Our next step will be to extend this recognition system for the connected digits recognition task with missing data, following the Aurora standard for robust ASR (Pearce et al. 2000). This will provide a direct comparison with HMM-based missing data recognition (Barker et al., 2001). In this case we will need to introduce 'silence' as an additional recognition class, and the training targets will be obtained by forced-alignment on clean speech with an existing recogniser. We will use realistic missing data masks, rather than random deletions. This is known to be a more demanding condition (Cooke et al. 1996).

When we are training using clean speech with added noise, another possibility is to use the true values of the corrupted features as training targets for imputation. Use of actual targets for missing values has been reported by (Seung, 1997) but the RNN architecture in the latter work supports only pattern completion.

### Acknowledgement

This work is being supported by Nokia Mobile Phones, Denmark and the UK Overseas Research Studentship scheme.

### References

Ahmed, S. & Tresp, V. (1993). Some solutions to the missing feature problem in vision. Advances in

Neural Information Processing Systems 5 (S.J.Hanson, J.D.Cowan & C.L.Giles, eds.), Morgan Kaufmann, San Mateo, CA, 393-400.

Barker, J., Green, P.D. and Cooke, M.P. (2001). Linking auditory scene analysis and robust ASR by missing data techniques. Workshop on Innovation in Speech Processing 2001, Stratford-upon-Avon, UK.

Barker, J., Josifovski, L., Cooke, M.P. and Green, P.D. (2000a). Soft decisions in missing data techniques for robust automatic speech recognition. Accepted for ICSLP-2000, Beijing.

Barker, J., Cooke, M.P. and Ellis, D.P.W. (2000b). Decoding speech in the presence of other sound sources. Accepted for ICSLP-2000, Beijing

Bourlard, H. and N. Morgan (1998). Hybrid HMM/ANN systems for speech recognition: Overview and new research directions. In C. L.Giles and M. Gori (Eds.), Adaptive Processing of Sequences and Data Structures, Volume 1387 of Lecture Notes in Artificial Intelligence, pp. 389--417. Springer.

Cooke, M., Green, P., Josifovski, L. and Vizinho, A. (2001). Robust automatic speech recognition with missing and unreliable acoustic data. submitted to Speech Communication, 24th June 1999.

Cooke, M.P., Morris, A. & Green, P.D. (1996). Recognising occluded speech. ESCA Tutorial and Workshop on the Auditory Basis of Speech Perception, Keele University, July 15-19.

Drygajlo, A. & El-Maliki, M. (1998). Speaker verification in noisy environment with combined spectral subtraction and missing data theory. Proc ICASSP-98, vol. I, pp121-124.

Elman, J.L. (1990). Finding structure in time. Cognitive Science, 14, 179-211.

Furui, S. (1997). Recent advances in robust speech recognition. Proc. ESCA-NATO Tutorial and Research Workshop on Robust Speech Recognition for Unknown Communication Channels, France, pp.11-20.

Gingras, F. and Bengio, Y. (1998). Handling Asynchronous or Missing Data with Recurrent Networks. International Journal of Computational Intelligence and Organizations, vol. 1, no. 3, pp. 154-163.

Ghahramani, Z. & Jordan, M.I. (1994). Supervised learning from incomplete data via an EM approach. Advances in Neural Information Processing Systems 6 (J.D. Cowan, G. Tesauro & J. Alspector, eds.), Morgan Kaufmann, San Mateo, CA, pp.120-129.

Jordan, M. I. (1998). Supervised learning and systems with excess degrees of freedom. Technical Report COINS TR 88-27, Massachusetts Institute of Technology, 1988.

Josifovski, L., Cooke, M., Green, P. and Vizinho, A. (1999). State based imputation of missing data for robust speech recognition and speech enhancement. Proc. Eurospeech'99, Budapest, Vol. 6, pp. 2837-2840.

Leonard, R. G., (1984). A Database for Speaker-Independent Digit Recognition. Proc. ICASSP 84, Vol. 3, p. 42.11, 1984.

Lippmann, R. P. (1997). Speech recognition by machines and humans. Speech Communication vol. 22 no. 1 pp. 1-15.

Morris, A., Josifovski, L., Bourlard, H., Cooke, M.P. and Green, P.D. (2000). A neural network for classification with incomplete data: application to robust ASR. ICSLP 2000, Beijing China.

Pearce, D. and Hirsch, H.--G. (2000). The aurora experimental framework for the performance evaluation of speech recognition systems under noisy conditions. In Proc. ICSLP 2000, IV, 29--32, Beijing, China.

Pedersen, M. W. (1997). Optimization of Recurrent Neural Networks for Time Series Modeling. PhD thesis. Technical University of Denmark.

Raj, B., Seltzer, M., & Stern, R. (2000). Reconstruction of damaged spectrographic features for robust speech recognition. ICSLP 2000.

Seung, H. S. (1997). Learning continuous attractors in Recurrent Networks. Proc. NIPS'97 pp 654-660.

Vizinho, A., Green, P., Cooke, M. and Josifovski, L. (1999). Missing data theory, spectral subtraction and signal-to-noise estimation for robust ASR: An integrated study. Proc. Eurospeech'99, Budapest, Sep. 1999, Vol. 5, pp. 2407-2410.
